# A Continuous Speech Recognition System Embedding MLP into HMM

**Hervé Bourlard**
Philips Research Laboratory
Av. van Becelaere 2, Box 8
B-1170 Brussels, Belgium

**Nelson Morgan**
Intl. Comp. Sc. Institute
1947 Center Street, Suite 600
Berkeley, CA 94704, USA

## ABSTRACT

We are developing a phoneme based, speaker-dependent continuous speech recognition system embedding a Multilayer Perceptron (MLP) (i.e., a feedforward Artificial Neural Network), into a Hidden Markov Model (HMM) approach. In [Bourlard & Wellekens], it was shown that MLPs were approximating Maximum a Posteriori (MAP) probabilities and could thus be embedded as an emission probability estimator in HMMs. By using contextual information from a sliding window on the input frames, we have been able to improve frame or phoneme classification performance over the corresponding performance for simple Maximum Likelihood (ML) or even MAP probabilities that are estimated without the benefit of context. However, recognition of words in continuous speech was not so simply improved by the use of an MLP, and several modifications of the original scheme were necessary for getting acceptable performance. It is shown here that word recognition performance for a simple discrete density HMM system appears to be somewhat better when MLP methods are used to estimate the emission probabilities.

## 1   INTRODUCTION

We have performed a number of experiments with a 1000-word vocabulary continuous speech recognition task. Our frame classification results [Bourlard et al., 1989]

are consistent with other research showing the capabilities of MLPs trained with back-propagation-styled learning schemes for the recognition of voiced-unvoiced speech segments [Gevins & Morgan, 1984], isolated phonemes [Watrous & Shastri, 1987; Waibel et al., 1988; Makino et al., 1983], or of isolated words [Peeling & Moore, 1988]. These results indicate that "neural network" approaches can, for some problems, perform pattern classification at least as well as traditional HMM approaches. However, this is not particularly mysterious. When traditional statistical assumptions (distribution, independence of multiple features, etc.) are not valid, systems which do not rely on these assumptions can work better (as discussed in [Niles et al., 1989]). Furthermore, networks provide an easy way to incorporate multiple sources of evidence (multiple features, contextual windows, etc.) without restrictive assumptions.

However, it is not so easy to improve the recognition of words in continuous speech by the use of an MLP. For instance, while it has been shown that the outputs of a feedforward network can be used as emission probabilities in an HMM [Bourlard et al., 1989], the corresponding word recognition performance can be very poor. This is true even when the same network demonstrates extremely good performance at the frame or phoneme levels. We have developed a hybrid MLP-HMM algorithm which (for a preliminary experiment) appears to exceed performance of the same HMM system using standard statistical approaches to estimate the emission probabilities. This was only possible after the original algorithm was modified in ways that did not necessarily maximize the frame recognition performance for the training set. We will describe these modifications below, along with experimental results.

## 2  METHODS

As shown by both theoretical [Bourlard & Wellekens, 1989] and experimental [Bourlard & Morgan, 1989] results, MLP output values may be considered to be good estimates of MAP probabilities for pattern classification. Either these, or some other related quantity (such as the output normalized by the prior probability of the corresponding class) may be used in a Viterbi search to determine the best time-warped succession of states (speech sounds) to explain the observed speech measurements. This hybrid approach (MLP to estimate probabilities, HMM to incorporate them to recognize continuous speech as a succession of words) has the potential of exploiting the interpolating capabilities of MLPs while using a Dynamic Time Warping (DTW) procedure to capture the dynamics of speech.

However, to achieve good performance at the word level, the following modifications of this basic scheme were necessary:

- MLP training methods - a new cross-validation [Stones, 1977] training algorithm was designed in which the stopping criterion was based on performance for an independent validation set [Morgan & Bourlard, 1990]. In other words, training was stopped when performance on a second set of data began going down, and not when training error leveled off. This greatly improved generalization, which could be further tested on a third independent validation set.

- probability estimation from the MLP outputs - In the original scheme [Bourlard & Wellekens, 1989], MLP outputs were used as MAP probabilities for the HMM directly. While this helped frame performance, it hurt word performance. This may have been due at least partly to a mismatch between the relative frequency of phonemes in the training sets and test (word recognition) sets. Division by the prior class probabilities as estimated from the training set removed this effect of the priors on the DTW. This led to a small decrease in frame classification performance, but a large (sometimes 10 - 20%) improvement in word recognition rates (see Table 1 and accompanying description).

- word transition costs for the underlying HMM - word transition penalties had to be increased for larger contextual windows to avoid a large number of insertions; see Section 4. This is shown to be equivalent to keeping the same word transition cost but scaling the log probabilities down by a number which reflected the dependence of neighboring frames. A reasonable value for this can be determined from recognition on a small number of sentences (e.g., 50), choosing a value which results in insertions at most equal to the number of deletions.

- segmentation of training data - much as with HMM systems, an iterative procedure was required to time align the training labels in a manner that was statistically consistent with the recognition methods used. In our most recent experiments, we segmented the data using an iterative Viterbi alignment starting from a segmentation based on average phoneme durations, and terminated at the segmentation which led to the best performance on an independent test set. For one of our speakers, we had available a more accurate frame labeling (produced by an automatic but more complex procedure [Aubert, 1987]) to use as a start point for the iteration, which led to even better performance.

## 3   EXPERIMENTAL APPROACH

We have been using a speaker-dependent German database (available from our collaboration with Philips) called SPICOS [Ney & Noll, 1988]. The speech had been sampled at a rate of 16 kHz, and 30 points of smoothed, "mel-scaled" logarithmic spectra (over bands from 200 to 6400 Hz) were calculated every 10-ms from a 512-point FFT over a 25-ms window. For our experiments, the mel spectrum and the energy were vector-quantized to pointers into a single speaker-dependent table of prototypes.

Two independent sets of vocabularies for training and test are used. The training dataset consists of two sessions of 100 German sentences per speaker. These sentences are representative of the phoneme distribution in the German language and include 2430 phonemes in each session. The two sessions of 100 sentences are phonetically segmented on the basis of 50 phonemes, using a fully automated procedure [Aubert, 1987]. The test set consists of one session of 200 sentences per speaker. The recognition vocabulary contains 918 words (including the "silence" word) and the overlap between training and recognition is 51 words. Most of the latter are articles, prepositions and other structural words. Thus, the training and test are essentially vocabulary-independent. Initial tests

used sentences from a single male speaker. The final algorithms were tested on an additional male and female speaker.

The acoustic vectors were coded on the basis of 132 prototype vectors by a simple binary representation with only one bit 'on'. Multiple frames were used as input to provide context to the network. In the experiments reported here, the context was 9 frames, while the size of the output layer was kept fixed at 50 units, corresponding to the 50 phonemes to be recognized. The input field contained $9 \times 132 = 1188$ units, and the total number of possible inputs was thus equal to $132^9$. There were 26767 training patterns (from the first training session of 100 sentences) and 26702 independent test patterns (from the second training session of 100 sentences). Of course, this represented only a very small fraction of the possible inputs, and generalization was thus potentially difficult. Training was done by the classical "error-back propagation" algorithm, starting by minimizing an entropy criterion, and then the standard least-mean-square error criterion. In each iteration, the complete training set was presented, and the parameters were updated after each training pattern. To avoid overtraining of the MLP, improvement on a cross-validation set was checked after each iteration and if classification was decreasing, the adaptation parameter of the gradient procedure was reduced, otherwise it was kept constant. Later on this approach was systematized by splitting the data in three parts: one for training, one for cross-validation and a third one absolutely independent of the training procedure for the actual validation. No significant difference was observed between classification rates for the last two data sets.

In [Bourlard et al., 1989] this procedure was shown yielding improved frame classification performance over simple ML and MAP estimates. However, acceptable word recognition perfomance was still difficult to reach.

## 4  WORD RECOGNITION RESULTS

The output values of the MLP were evaluated for each frame, and (after division by the prior probability of each phoneme) were used as emission probabilities in a discrete HMM system. In this system, each phoneme was modeled with a single conditional density, repeated $D/2$ times, where $D$ was a prior estimate of the duration of the phoneme. Only selfloops and sequential transitions were permitted. A Viterbi decoding was then used for recognition of the first hundred sentences of the test session (on which word entrance penalties were optimized), and our best results were validated by a further recognition on the second hundred sentences of the test set. Note that this same simplified HMM was used for both the ML reference system (estimating probabilities directly from relative frequencies) and the MLP system, and that the same input features were used for both.

Table 1 shows the recognition rate (100% - error rate, where errors includes insertions, deletions, and substitutions) for the first 100 sentences of the test session. All runs except the last were done with 20 hidden units in the MLP, as suggested by frame performance. Note the significant positive effect of division of the MLP outputs, which are trained to approximate MAP probabilities, by estimates of the prior probabilities for each class (denoted "MLP/priors" in Table 1).

**Table 1:** Word Recognition, speaker m003

| system method | size of context | % correct | |
|---|---|---|---|
| | | test | validation |
| MLP | 1 | 27.3 | |
| MLP/priors | 1 | 49.7 | |
| MLP | 9 | 40.9 | |
| MLP/priors | 9 | 51.9 | 52.2 |
| ML | 1 | 52.6 | 52.5 |
| MLP/priors (0 hidden) | 9 | 53.3 | |

**Table 2:** Word Recognition using Viterbi segmentation, speaker m003

| method | context | test |
|---|---|---|
| MLP/priors (0 hidden) | 9 | 65.3 |
| ML | 1 | 56.9 |

Word transition probabilities were optimized for both the Maximum Likelihood and MLP style HMMs. This led to a word exit probability of $10^{-8}$ for the ML and for 1-frame MLP's, and $10^{-14}$ for an MLP with 9 frames of context. After these adjustments, performance was essentially the same for the two approaches. Performance on the last hundred sentence of the test session (shown in the last column of Table 1) validated that the two systems generalized equivalently despite these tunings.

An initial time alignment of the phonetic transcription with the data (for this speaker) had previously been calculated using a program incorporating speech-specific knowledge [Aubert, 1987]. This labeling had been used for the targets of the frame-based training described above. We then used this alignment as a "bootstrap" segmentation for an iterative Viterbi procedure, much as is done in conventional HMM systems. As with the MLP training, the data was divided into a training and cross-validation set, and the best segmentation (corresponding to the best validation set frame classification rate) was used for later training. For both cross-validation procedures, we switched to a training set of 150 sentences (two repetitions of 75 sentences) and a cross-validation set of 50 sentences (two repetitions of 25 each). Finally, since the best performance in Table 1 was achieved using no hidden layer, we continued our experiments using this simpler network, which also required only a simple training procedure (entropy error criterion only). Table 2 shows this performance for the full 200 recognition sentences (test + validation sets from Table 1).

Two of the more puzzling observations in this work were the need to increase word entrance penalties with the width of the input context and the difficulty to reflect good frame performance at the word level. MLPs can make better frame level discriminations

than simple statistical classifiers, because they can easily incorporate multiple sources of evidence (multiple frames, multiple features) without simplifying assumptions. However, when the input features within a contextual window are roughly independent, the Viterbi algorithm will already incorporate all of the context in choosing the best HMM state sequence explaining an utterance. If emission probabilities are estimated from the outputs of an MLP which has a $2c + 1$ frame contextual input, the probability to observe a feature sequence $\{f_1, f_2, \ldots, f_N\}$ (where $f_n$ represents the feature vector at time $n$) on a particular HMM state $q_k$ is estimated as:

$$\prod_{i=1}^{N} p(f_{i-c}, \ldots, f_i, \ldots, f_{i+c}|q_k),$$

where Bayes' rule has already been used to convert the MLP outputs (which estimate MAP probabilities) into ML probabilities. If independence is assumed, and if boundary effects (context extending before frame 1 or after frame N) are ignored (assume $(2c+1) \ll N$), this becomes:

$$\prod_{i=1}^{N} \prod_{j=-c}^{c} p(f_{i+j}|q_k) = \prod_{i=1}^{N} [p(f_i|q_k)]^{2c+1},$$

where the latter probability is just the classical Maximum Likelihood solution, raised to the power $2c + 1$. Thus, if the features are independent over time, to keep the effect of transition costs the same as for the simple HMM, the log probabilities must be scaled down by the size of the contextual window. Note that, in the more realistic case where dependencies exist between frames, the optimal scaling factor will be less than $2c + 1$, down to a minimum of 1 for the case in which frames are completely dependent (e.g., same within a constant factor); the scaling factor should thus reflect the time correlation of the input features. Thus, if the features are assumed independent over time, there is no advantage to be gained by using an MLP to extract contextual information for the estimation of emission probabilities for an HMM Viterbi decoding. In general, the relation between the MLP and ML solutions will be more complex, because of interdependence over time of the input features. However, the above relation may give some insight as to the difficulty we have met in improving word recognition performance with a single discrete feature (despite large improvements at the frame level). More positively, our results show that the probabilities estimated by MLPs can be used at least as effectively as conventional estimates and that some advantage can be gained by providing more information for estimating these probabilities.

We have duplicated our recognition tests for two other speakers from the same data base. In this case, we labeled each training set (from the original male plus a male and a female speaker) using a Viterbi iteration initialized from a time-alignment based on a simple estimate of average phoneme duration. This reduced all of the recognition scores, underlining the necessity of a good start point for the Viterbi iteration. However, as can be seen from the Table 3 results (measured over the full 200 recognition sentences), the MLP-based methods appear to consistently offer at least some measurable improvement over the simpler estimation technique. In particular, the performance for the two systems differed significantly ($p < 0.001$) for two out of three speakers, as well as for a multispeaker

**Table 3:** Word Recognition for 3 speakers, simple initialization

| speaker | MLE | MLP |
|---------|-----|-----|
| m003 | 54.4 | 59.7 |
| m001 | 47.4 | 51.9 |
| w010 | 54.2 | 54.3 |

comparison over the three speakers (in each case using a normal approximation to a binomial distribution for the null hypothesis).

## 5    CONCLUSION

These results show some of the improvement for MLPs over conventional HMMs which one might expect from the frame level results. MLPs can sometimes make better frame level discriminations than simple statistical classifiers, because they can easily incorporate multiple sources of evidence (multiple frames, multiple features), which is difficult to do in HMMs without major simplifying assumptions. In general, the relation between the MLP and ML word recognition is more complex. Part of the difficulty with good recognition may be due to our choice of discrete, vector-quantized features, for which no metric is defined over the prototype space. Despite these limitations, it now appears that the probabilities estimated by MLPs may offer improved word recognition through the incorporation of context in the estimation of emission probabilities. Furthermore, our new result shows the effectiveness of Viterbi segmentation in labeling training data for an MLP. This result appears to remove a major handicap of MLP use, i.e. the requirement for hand-labeled speech, and also offers the possibility to deal with more complex HMMs.

### Acknowledgments

Support from the International Computer Science Institute (ICSI) and Philips Research for this work is gratefully acknowledged. Chuck Wooters of ICSI and UCB provided much-needed assistance, and Xavier Aubert of Philips put together our Spicos materials.

### References

X. Aubert, (1988), "Supervised Segmentation with Application to Speech Recognition", in *Proc. Eur. Conf. Speech Technology*, Edinburgh, p.161-164.

H. Bourlard, N. Morgan, & C.J. Wellekens, (1989), "Statistical Inference in Multilayer Perceptrons and Hidden Markov Models with Applications in Continuous Speech Recognition", to appear in *Neuro Computing, Algorithms, and Applications*, NATO ASI Series.

H. Bourlard, H. & N. Morgan, (1989), "Merging Multilayer Perceptrons and Hidden Markov Models: Some Experiments in Continuous Speech Recognition" International Computer Science Institute TR-89-033.

H. Bourlard & C.J. Wellekens, (1989), "Links between Markov models and multilayer perceptrons", to be published in *IEEE Trans. on Pattern Analysis and Machine Intelligence*, 1990.

A. Gevins & N. Morgan, (1984), "Ignorance-Based Systems", *Proc. IEEE Intl. Conf. on Acoustics, Speech, & Signal Processing*, Vol. 3, 39A5.1-39A5.4, San Diego.

S. Makino, T. Kawabata, T. & K. Kido, (1983), "Recognition of consonants based on the Perceptron Model", *Proc. IEEE Intl. Conf. on Acoustics, Speech, & Signal Processing*, Vol. 2, pp. 738-741, Boston, Mass.

N. Morgan & H. Bourlard, (1989), "Generalization and Parameter Estimation in Feedforward Nets: Some Experiments", *Advances in Neural Information Processing Systems II*, Morgan Kaufmann.

H. Ney & A. Noll, (1988), "Phoneme Modeling Using Continuous Mixture Densities", *Proc. IEEE Intl. Conf. on Acoustics, Speech, & Signal Processing*, Vol. 1, pp. 437-440, New York.

L. Niles, H. Silverman, G. Tajchman & M. Bush, (1989), "How Limited Training Data Can Allow a Neural Network Classifier to Outperform an 'Optimal' Statistical Classifier", *Proc. IEEE Intl. Conf. on Acoustics, Speech, & Signal Processing*, Vol. 1, pp. 17-20, Glasgow, Scotland.

S.M. Peeling, S.M. & R.K. Moore, (1988), "Experiments in Isolated Digit Recognition Using the Multi-Layer Perceptron", Royal Speech and Radar Establishment, Technical Report 4073, Malvern, Worcester.

M. Stone, (1987), "Cross-validation: a review", *Math. Operationforsch. Statist. Ser. Statist.*, vol.9, pp. 127-139.

A. Waibel, T. Hanazawa, G. Hinton, K. Shikano & K. Lang, (1988), "Phoneme Recognition: Neural Networks vs. Hidden Markov Models", *Proc. IEEE Intl. Conf. on Acoustics, Speech, & Signal Processing*, Vol. 1, pp. 107-110, New York.

R. Watrous & L. Shastri, (1987), Learning phonetic features using connectionist networks: an experiment in speech recognition", *Proceedings of the First Intl. Conference on Neural Networks*, IV-381-388, San Diego, CA.
